# Gaussian Process Preference Elicitation

**Edwin V. Bonilla,   Shengbo Guo,   Scott Sanner**
NICTA & ANU, Locked Bag 8001, Canberra ACT 2601, Australia
{edwin.bonilla, shengbo.guo, scott.sanner}@nicta.com.au

## Abstract

Bayesian approaches to preference elicitation (PE) are particularly attractive due to their ability to explicitly model uncertainty in users' latent utility functions. However, previous approaches to Bayesian PE have ignored the important problem of generalizing from previous users to an *unseen* user in order to reduce the elicitation burden on new users. In this paper, we address this deficiency by introducing a Gaussian Process (GP) prior over users' latent utility functions on the *joint* space of user and item features. We learn the hyper-parameters of this GP on a set of preferences of previous users and use it to aid in the elicitation process for a new user. This approach provides a flexible model of a multi-user utility function, facilitates an efficient value of information (VOI) heuristic query selection strategy, and provides a principled way to incorporate the elicitations of multiple users back into the model. We show the effectiveness of our method in comparison to previous work on a real dataset of user preferences over sushi types.

## 1   Introduction

Preference elicitation (PE) is an important component of interactive decision support systems that aim to make optimal recommendations to users by actively querying their preferences. A crucial requirement for PE systems is that they should be able to make optimal or near optimal recommendations based only on a small number of queries. In order to achieve this, a PE system should (a) maintain a flexible representation of the user's utility function; (b) handle uncertainty in a principled manner; (c) select queries that allow the system to discriminate amongst the highest utility items; and (d) allow for the incorporation of prior knowledge from different sources.

While previous Bayesian PE approaches have addressed (a), (b) and (c), they appear to ignore an important aspect of (d) concerning generalization from previous users to a *new* unseen user in order to reduce the elicitation burden on new users. In this paper we propose a Bayesian PE approach to address (a)–(d), including generalization to new users, in an elegant and principled way. Our approach places a (correlated) Gaussian process (GP) prior over the latent utility functions on the joint space of user features ($\mathcal{T}$, mnemonic for tasks) and item features ($\mathcal{X}$). User preferences over items are then seen as drawn from the comparison of these utility function values.

The main advantages of our GP-based Bayesian PE approach are as follows. First, due to the non-parametric Bayesian nature of GPs, we have a *flexible* model of the user's utility function that can *handle uncertainty* and incorporate evidence straightforwardly. Second, by having a GP over the joint $\mathcal{T} \times \mathcal{X}$ space, we can integrate *prior knowledge* on user similarity or item similarity, or simply have more general-purpose covariances whose parameterization can be learned from observed preferences of previous users (i.e. achieving integration of multi-user information). Finally, our approach draws from concepts in the Gaussian process optimization and decision-making literature [1, 2] to propose a Bayesian decision-theoretic PE approach. Here the required *expected value of information* computations can be derived in closed-form to facilitate the selection of informative queries and determine the highest utility item from the available item set as quickly as possible.

In this paper we focus on pairwise comparison queries for PE, which are known to have low cognitive load [3, 4]. In particular, we assume a likelihood model of pairwise preferences that factorizes over users and preferences and a GP prior over the latent utility functions correlates users and items.

## 2 Problem Formulation

Let $x$ denote a specific item (or product) that is described by a set of features $\mathbf{x}$ and $t$ denote a user (mnemonic for task) that can be characterized with features $\mathbf{t}$. For a set of items $\mathbb{X} = \{x_1, \ldots, x_N\}$ and users $\mathbb{T} = \{t_1, \ldots, t_M\}$ we are given a set of training preference pairs:

$$\mathcal{D} = \left\{ (t^{(j)}, x_{k_1}^{(j)} \succ x_{k_2}^{(j)}) | k = 1, \ldots, K_j; k_1, k_2 \in \{1, \ldots, N\}; j = 1 \ldots, M \right\}, \tag{1}$$

where $x_{k_1}^{(j)} \succ x_{k_2}^{(j)}$ denotes that we have observed that user $j$ prefers item $k_1$ over item $k_2$ and $K_j$ is the number of preference relations observed for user $j$.

The preference elicitation problem is that given a new user, described by a set of features $\mathbf{t}^*$, we aim to determine (or elicit) what his/her preferences (or favourite items) are by asking a small number of queries of the form $q_{ij} \stackrel{\text{def}}{=} x_i \succ x_j$, meaning that he/she will prefer item $i$ over item $j$. Ideally, we would like to obtain the best user preferences with the smallest number of possible queries.

The key idea of this paper is that of learning a Gaussian process (GP) model over users' latent utility functions and use this model in order to drive the elicitation process of a new user. Due to the non-parametric Bayesian nature of the GPs, this allows us to have a powerful model of the user's utility function and to incorporate the evidence (i.e. the responses the user gives to our queries) in a principled manner. Our approach directly exploits: (a) user-relatedness, i.e. that users with similar characteristics may have similar preferences; (b) items' similarities and (c) the value of information of obtaining a response to a query in order to elicit the preferences of the user.

## 3 Likelihood Model

Our likelihood model considers that the users' preference relationships are conditionally independent given the latent utility functions. In other words, the probability of a user $t$ preferring item $x$ over item $x'$ given their utility functions is:

$$p(x^t \succ x'^t | f(\mathbf{t}, \mathbf{x}), f(\mathbf{t}, \mathbf{x}'), \xi) = \mathbb{I}[f(\mathbf{t}, \mathbf{x}) - f(\mathbf{t}, \mathbf{x}') \geq \xi] \quad \text{with} \quad p(\xi) = \mathcal{N}(\xi | 0, \sigma^2), \tag{2}$$

where $\mathbb{I}[\cdot]$ is an indicator function that is 1 if the condition $[\cdot]$ is true and 0 otherwise; and $\sigma^2$ is the variance of the normally distributed variable $\xi$ that dictates how different the latent functions should be for the corresponding relation to hold. Hence:

$$p(x^t \succ x'^t | f(\mathbf{t}, \mathbf{x}), f(\mathbf{t}, \mathbf{x}')) = \int_{-\infty}^{\infty} \mathbb{I}[f(\mathbf{t}, \mathbf{x}) - f(\mathbf{t}, \mathbf{x}') \geq \xi] \mathcal{N}(\xi | 0, \sigma^2) d\xi \tag{3}$$

$$= \Phi \left( \frac{f(\mathbf{t}, \mathbf{x}) - f(\mathbf{t}, \mathbf{x}')}{\sigma} \right), \tag{4}$$

where $\Phi(\cdot)$ is the Normal cumulative distribution function (cdf). The conditional data-likelihood is then given by:

$$p(\mathcal{D}|\mathbf{f}) = \prod_{j=1}^{M} \prod_{k=1}^{K_j} \Phi(z_k^j) \quad \text{with} \quad z_k^j = \frac{1}{\sigma} \left( f(\mathbf{t}^{(j)}, \mathbf{x}_{k_1}^{(j)}) - f(\mathbf{t}^{(j)}, \mathbf{x}_{k_2}^{(j)}) \right). \tag{5}$$

## 4 Modeling User Dependencies with a GP Prior

As mentioned above, we model user (and item) dependencies via the user latent utility functions, which are assumed to be drawn from a GP prior that accounts for user similarity and item similarity directly:

$$f(\mathbf{t}, \mathbf{x}) \sim \mathcal{GP} \left( \mathbf{0}, \kappa^t(\mathbf{t}, \mathbf{t}') \kappa^x(\mathbf{x}, \mathbf{x}') \right), \tag{6}$$

where $\kappa^t(\cdot, \cdot)$ is a covariance function on user-descriptors $\mathbf{t}$ and $\kappa^x(\cdot, \cdot)$ is a covariance function on item features $\mathbf{x}$. We will denote the parameters of these covariance functions (so-called hyper-parameters) by $\boldsymbol{\theta}^t$ and $\boldsymbol{\theta}^x$. (These types of priors have been considered previously in the regression setting, see e.g. [5].)

Additionally, let $\mathbf{f}$ be the utility function values for all training users at all training input locations (i.e. items) so that $\mathbf{f} = [f(\mathbf{t}^{(1)}, \mathbf{x}^{(1)}), \dots f(\mathbf{t}^{(1)}, \mathbf{x}^{(N)}), \dots, f(\mathbf{t}^{(M)}, \mathbf{x}^{(1)}), \dots, f(\mathbf{t}^{(M)}, \mathbf{x}^{(N)})]^T$ and $\mathbf{F}$ be the $N \times M$ matrix for which the $j$th column corresponds to the latent values for the $j$th user at all input points such that $\mathbf{f} = \text{vec}\,\mathbf{F}$. Hence:

$$\mathbf{f} \sim \mathcal{N}(\mathbf{0}, \boldsymbol{\Sigma}) \quad \text{with} \quad \boldsymbol{\Sigma} = \mathbf{K}^t \otimes \mathbf{K}^x, \tag{7}$$

where $\mathbf{K}^t$ is the covariance between all the training users, $\mathbf{K}^x$ is the covariance between all the training input locations, and $\otimes$ denotes the Kronecker product. Note that dependencies between users are not arbitrarily imposed but rather they will be learned from the available data by optimizing the marginal likelihood. (We will describe the details of hyper-parameter learning in section 7.)

## 5  Posterior and Predictive Distributions

Given the data in (1) and the prior over the latent utility functions in equation (6), we can obtain the posterior distribution:

$$P(\mathbf{f}|\mathcal{D}, \boldsymbol{\theta}) = \frac{p(\mathcal{D}|\mathbf{f}, \boldsymbol{\theta})p(\mathbf{f}|\boldsymbol{\theta})}{p(\mathcal{D}|\boldsymbol{\theta})}, \tag{8}$$

where we have emphasized the dependency on the hyper-parameters $\boldsymbol{\theta}$ that include $\boldsymbol{\theta}^t$, $\boldsymbol{\theta}^x$ and $\sigma^2$ and where $p(\mathcal{D}|\boldsymbol{\theta})$ is the marginal likelihood (or evidence) with $p(\mathcal{D}|\boldsymbol{\theta}) = \int p(\mathcal{D}|\mathbf{f}, \boldsymbol{\theta})p(\mathbf{f}|\boldsymbol{\theta})d\mathbf{f}$. The non-Gaussian nature of the conditional likelihood term (given in equation (5)) makes the above integral analytically intractable and hence we will require approximations. In this paper we will focus on analytical approximations and more specifically, we will approximate the posterior $p(\mathbf{f}|\mathcal{D}, \boldsymbol{\theta})$, and the evidence, using the Laplace approximation.

The Laplace method approximates the true posterior with a Gaussian: $p(\mathbf{f}|\mathcal{D}, \boldsymbol{\theta}) \approx \mathcal{N}(\mathbf{f}|\hat{\mathbf{f}}, \mathbf{A}^{-1})$, where $\hat{\mathbf{f}} = \text{argmax}_{\mathbf{f}}\, p(\mathbf{f}|\mathcal{D}, \boldsymbol{\theta}) = \text{argmax}_{\mathbf{f}}\, p(\mathcal{D}|\mathbf{f}, \boldsymbol{\theta})p(\mathbf{f}|\boldsymbol{\theta})$ and $\mathbf{A}$ is the Hessian of the negative log-posterior evaluated at $\hat{\mathbf{f}}$. Hence we consider the unnormalized expression $p(\mathcal{D}|\mathbf{f}, \boldsymbol{\theta})p(\mathbf{f}|\boldsymbol{\theta})$ and, omitting the terms that are independent of $\mathbf{f}$, we focus on the maximization of the following expression:

$$\psi(\mathbf{f}) = \sum_{j=1}^{M} \sum_{k=1}^{K_j} \log \Phi(z_k^j) - \frac{1}{2}\mathbf{f}^T \boldsymbol{\Sigma}^{-1} \mathbf{f}. \tag{9}$$

Using Newton's method we obtain the following iterative update:

$$\mathbf{f}^{\text{new}} = (\mathbf{W} + \boldsymbol{\Sigma}^{-1})^{-1} \left( \frac{\partial \log p(\mathcal{D}|\mathbf{f}, \boldsymbol{\theta})}{\partial \mathbf{f}} + \mathbf{W}\mathbf{f} \right) \text{ with } \mathbf{W}_{pq} = -\sum_{j=1}^{M} \sum_{k=1}^{K_j} \frac{\partial^2 \log \Phi(z_k^j)}{\partial f_p \partial f_q}. \tag{10}$$

Once we have found the maximum posterior $\hat{\mathbf{f}}$ by using the above iteration we can show that:

$$p(\mathbf{f}|\mathcal{D}) \approx \mathcal{N}(\mathbf{f}|\hat{\mathbf{f}}, (\mathbf{W} + \boldsymbol{\Sigma}^{-1})^{-1}). \tag{11}$$

### 5.1  Predictive Distribution

In order to set-up our elicitation framework we will also need the predictive distribution for a fixed test user $\mathbf{t}_*$ at an unseen pair $\mathbf{x}_*^1$, $\mathbf{x}_*^2$. This is given by:

$$p(\mathbf{f}_*|\mathcal{D}) = \int p(\mathbf{f}_*|\mathbf{f})p(\mathbf{f}|\mathcal{D})d\mathbf{f} \tag{12}$$

$$= \mathcal{N}(\mathbf{f}_*|\boldsymbol{\mu}_*, \mathbf{C}_*), \tag{13}$$

with:

$$\boldsymbol{\mu}_* = \mathbf{k}_*^T \boldsymbol{\Sigma}^{-1} \hat{\mathbf{f}} \quad \text{and} \quad \mathbf{C}_* = \boldsymbol{\Sigma}_* - \mathbf{k}_*^T (\boldsymbol{\Sigma} + \mathbf{W}^{-1})^{-1} \mathbf{k}_*, \tag{14}$$

where $\boldsymbol{\Sigma}$ is defined as in equation (7) and:

$$\mathbf{k}_* = \mathbf{k}_*^t \otimes \mathbf{k}_*^x \tag{15}$$

$$\mathbf{k}_*^t = \begin{bmatrix} \kappa^t(\mathbf{t}_*, \mathbf{t}^{(1)}), & \dots & \kappa^t(\mathbf{t}_*, \mathbf{t}^{(M)}) \end{bmatrix}^T \tag{16}$$

$$\mathbf{k}_*^x = \begin{bmatrix} \kappa^x(\mathbf{x}_*^1, \mathbf{x}^{(1)}), & \dots & \kappa^x(\mathbf{x}_*^1, \mathbf{x}^{(N)}) \\ \kappa^x(\mathbf{x}_*^2, \mathbf{x}^{(1)}), & \dots & \kappa^x(\mathbf{x}_*^2, \mathbf{x}^{(N)}) \end{bmatrix}^T \tag{17}$$

$$\boldsymbol{\Sigma}_* = \begin{bmatrix} \kappa^t(\mathbf{t}_*, \mathbf{t}_*)\kappa^x(\mathbf{x}_*^1, \mathbf{x}_*^1) & \kappa^t(\mathbf{t}_*, \mathbf{t}_*)\kappa^x(\mathbf{x}_*^1, \mathbf{x}_*^2) \\ \kappa^t(\mathbf{t}_*, \mathbf{t}_*)\kappa^x(\mathbf{x}_*^2, \mathbf{x}_*^1) & \kappa^t(\mathbf{t}_*, \mathbf{t}_*)\kappa^x(\mathbf{x}_*^2, \mathbf{x}_*^2) \end{bmatrix}. \tag{18}$$

## 6 Gaussian Process Preference Elicitation Framework

Now we have the main components to set up our preference elicitation framework for a test user characterized by features $\mathbf{t}_*$. Our main objective is to use the previously seen data (and the corresponding learned hyper-parameters) in order to drive the elicitation process and to incorporate the information obtained from the user's responses back into our model in a principled manner. Our main requirement is a function that dictates the value of making a query $q_{ij}$. In other words, we aim at trading-off the expected actual utility of the items involved in the query and the information these items will provide regarding the user's preferences. This is the exploration-exploitation dilemma, usually seen in optimization and reinforcement learning problems. We can address this issue by computing the expected value of information (EVOI, [2]) of making a query involving items $i$ and $j$. Before defining the EVOI, we will make use of the concept of expected improvement, a measure that is commonly used in optimization methods based on response surfaces (see e.g. [1]).

### 6.1 Expected Improvement

We have seen in equation (13) that the predictive distribution for the utility function on a test user $\mathbf{t}_*$ on item $\mathbf{x}$ follows a Gaussian distribution:

$$f(\mathbf{t}_*, \mathbf{x}|\mathcal{D}, \boldsymbol{\theta}) \sim \mathcal{N}(\mu_*(\mathbf{t}_*, \mathbf{x}), s_*^2(\mathbf{t}_*, \mathbf{x})), \tag{19}$$

where $\mu_*(\mathbf{t}_*, \mathbf{x})$ and $s_*^2(\mathbf{t}_*, \mathbf{x})$ can be obtained by using (the marginalized version of) equation (14). Let us assume that, at any point during the elicitation process we have an estimate of the utility of the best item and let us denote it by $f^{\text{best}}$. If we define the predicted improvement at $\mathbf{x}$ as $\mathcal{I} = f(\mathbf{t}_*, \mathbf{x}|\mathcal{D}, \boldsymbol{\theta}) - f^{\text{best}}$ then the expected improvement (EI) of recommending item $x$ (for a fixed user $\mathbf{t}_*$) instead of recommending the best item $x^{\text{best}}$ is given by:

$$\text{EI}(x|\mathcal{D}) = \int_0^\infty \mathcal{I} p(\mathcal{I}) d\mathcal{I} = s_*(\mathbf{t}_*, \mathbf{x})[z'\Phi(z') + \phi(z')], \tag{20}$$

where $z' = (\mu_*(\mathbf{t}_*, \mathbf{x}) - f^{\text{best}})/s_*(\mathbf{t}_*, \mathbf{x})$, $\Phi(\cdot)$ is the Normal cumulative distribution function (cdf) and $\phi(\cdot)$ is the Normal probability density function (pdf). Note that, for simplicity in the notation, we have omitted the dependency of $\text{EI}(x|\mathcal{D})$ on the user's features $\mathbf{t}_*$. Hence the maximum expected improvement (ME) under the current observed data $\mathcal{D}$ is:

$$\text{MEI}(\mathcal{D}) = \max_x \text{EI}(x|\mathcal{D}). \tag{21}$$

### 6.2 Expected Value of Information

Now we can define the expected value of information (EVOI) as the expected gain in improvement that is obtained by adding a query involving a particular pairwise relation. Thus, the expected value of information of obtaining the response for the queries involving items $x^{*i}$, $x^{*j}$ with corresponding utility values $\mathbf{f}^* = (f^*(\mathbf{t}_*, \mathbf{x}_*^i), f^*(\mathbf{t}_*, \mathbf{x}_*^j))^T$ is given by:

$$\text{EVOI}(\mathcal{D}, i, j) = -\text{MEI}(\mathcal{D}) + \left\langle \sum_{q_{ij}} p(q_{ij}|\mathbf{f}^*, \mathcal{D})\text{MEI}(\mathcal{D} \cup q_{ij}) \right\rangle_{p(\mathbf{f}^*|\mathcal{D})} \tag{22}$$

$$= -\text{MEI}(\mathcal{D}) + \left\langle p(x^{*i} \succ x^{*j}|\mathbf{f}^*, \mathcal{D}) \right\rangle_{p(\mathbf{f}^*|\mathcal{D})} \text{MEI}(\mathcal{D} \cup \{x^{*i} \succ x^{*j}\})$$

$$+ \left\langle p(x^{*j} \succ x^{*i}|\mathbf{f}^*, \mathcal{D}) \right\rangle_{p(\mathbf{f}^*|\mathcal{D})} \text{MEI}(\mathcal{D} \cup \{x^{*j} \succ x^{*i}\}), \tag{23}$$

---

**Algorithm 1** Gaussian Process Preference Elicitation

---

**Require:** hyper-parameters $\boldsymbol{\theta}^x, \boldsymbol{\theta}^t, \boldsymbol{\theta}^\sigma$ {learned from $M$ previous users} and corresponding $\mathcal{D}$
  **repeat**
    **for all** candidate pairs $(i, j)$ **do**
      Compute $\text{EVOI}(i, j, \mathcal{D}, \hat{\mathbf{f}}, \mathbf{W})$ {equation (23)}
    **end for**
    $(i^*, j^*) \leftarrow \text{argmax}_{i,j}\, \text{EVOI}(i, j)$ {best pair}
    Remove $(i^*, j^*)$ from candidate list
    **if** $q_{i^*, j^*}$ is true **then** {ask user and set true preference}
      $(i^{true}, j^{true}) \leftarrow (i^*, j^*)$
    **else**
      $(i^{true}, j^{true}) \leftarrow (j^*, i^*)$
    **end if**
    $\mathcal{D} \leftarrow \mathcal{D} \cup (t^{M+1}, x_{i^{true}} \succ x_{j^{true}})$ {Expand $\mathcal{D}$ and get $\mathcal{D}^+$}
    Update $\hat{\mathbf{f}}, \mathbf{W}$ {i.e. $P(\mathbf{f}|\mathcal{D})$ as in equation (10)}
  **until** Satisfied

---

where

$$\left\langle p(x^{*i} \succ x^{*j}|\mathbf{f}^*, \mathcal{D}) \right\rangle_{p(\mathbf{f}^*|\mathcal{D})} = p(x^{*i} \succ x^{*j}|\mathcal{D}) \tag{24}$$

$$= \int_{\mathbf{f}^*} p(x^{*i} \succ x^{*j}|\mathbf{f}^*, \mathcal{D}) p(\mathbf{f}^*|\mathcal{D}) d\mathbf{f}^* \tag{25}$$

$$= \int_{\mathbf{f}^*} \int_\xi \mathbb{I}[f_i^* - f_j^* \geq \xi] \mathcal{N}(\xi|0, \sigma^2) \mathcal{N}(\mathbf{f}_*|\boldsymbol{\mu}_*, \mathbf{C}_*) d\xi d\mathbf{f}^* \tag{26}$$

$$= \Phi\left( \frac{\mu_i^* - \mu_j^*}{\mathrm{C}_{i,i} - \mathrm{C}_{j,j} - 2\mathrm{C}_{i,j} - \sigma^2} \right), \tag{27}$$

and $\boldsymbol{\mu}_*$ and $\mathbf{C}_*$ as defined in (14). Note that in our model $p(x^{*j} \succ x^{*i}|\mathcal{D}) = 1 - p(x^{*i} \succ x^{*j}|\mathcal{D})$.

As mentioned above, $f^{\text{best}}$ can be thought of as an estimate of the utility of the best item as its true utility is unknown. In practice we maintain our beliefs over the utilities of the items $p(\mathbf{f}|\mathcal{D}^+)$ for the training users and the test user, where $\mathcal{D}^+$ denotes the data extended by the set of seen relationships on the test user (which is initially empty). Hence, we can set-up $f^{\text{best}} = \max_i \widehat{\mathbf{F}^+}_{i, M+1}$, where $\widehat{\mathbf{F}^+}$ is the matrix containing the mean estimates of the latent utility function distribution given by the Laplace approximation in equation (9). Alternatively, we can draw samples from such a distribution and apply the `max` operator.

In order to elicit preferences on a new user we simply select a query so that it maximizes the expected value of information EVOI as defined in equation (23). A summary of our approach is presented in algorithm 1. We note that although, in principle, one could also update the hyper-parameters based on the data provided by the new user, we avoid this in order to keep computations manageable at query time. The reasoning being that, implicitly, we have learned the utility functions over all users and we represent the utility of the test user (explicitly) on demand, updating our beliefs to incorporate the information provided by the user's responses.

## 7 Hyper-parameter Learning

Throughout this paper we have assumed that we have learned a Gaussian process model for the utility functions over users and items based upon previously seen preference relations. We refer to the hyper-parameters of our model as the hyper-parameters $\boldsymbol{\theta}^t$ and $\boldsymbol{\theta}^x$ of the covariance functions ($\kappa^t$ and $\kappa^x$ respectively) and $\boldsymbol{\theta}^\sigma = \log \sigma$, where $\sigma^2$ is the "noise" variance.

Although it is entirely possible to use prior knowledge on what these covariance functions are (or their corresponding parameter settings) for the specific problem under consideration, in many practical applications such prior knowledge is not available and one requires to tune such parameteriza-

tion based upon the available data. Fortunately, as in the standard GP regression framework, we can achieve this in a principled way through maximization of the marginal likelihood (or evidence).

As in the case of the posterior distribution, the marginal likelihood is analytically intractable and approximations are needed. The Laplace approximation to the marginal log-likelihood is given by:

$$\log p(\mathcal{D}|\boldsymbol{\theta}) \approx -\frac{1}{2}\log|\boldsymbol{\Sigma}\mathbf{W} + \mathbf{I}| - \frac{1}{2}\hat{\mathbf{f}}^T\boldsymbol{\Sigma}^{-1}\hat{\mathbf{f}} + \sum_{j=1}^{M}\sum_{k=1}^{K_j}\log\Phi(\hat{z}_k^j) \tag{28}$$

where $\hat{z}_k^j = z_k^j|_{\hat{\mathbf{f}}}$, $\hat{\mathbf{f}}$ and $\mathbf{W}$ are defined as in (10) and $\boldsymbol{\Sigma}$ is defined as in equation (7). Note that computations are not carried out at all the $M \times N$ data-points but only at those locations that "support" the seen relations and hence we should write e.g. $\hat{\mathbf{f}}_o$, $\boldsymbol{\Sigma}_o$ where the subindex $\{\}_o$ indicates this fact. However, for simplicity, we have omitted this notation.

Given the equation above, gradient-based optimization can be used for learning the hyper-parameters in our model. As we shall see in the following section, for our experiments we do not have much prior information on suitable hyper-parameter settings and therefore we have carried out hyper-parameter learning by maximization of the marginal log-likelihood.

# 8    Experiments & Results

In this section we describe the dataset used in our experiments, the evaluation setting and the results obtained with our model and other baseline methods.

## 8.1    The Sushi Dataset

We evaluate our approach on the Sushi dataset [6]. Here we present a brief description of this dataset and the pre-processing we have carried out in order to apply our method. The reader is referred to [6] for more details. The Sushi dataset contains *full* rankings given by 5000 Japanese users over $N = 10$ different types of sushi. Each sushi is associated with a set of features which include style, major group, minor group, heaviness, consumption frequency, normalized price and sell frequency. The first three features are categorical and therefore we have created the corresponding dummy variables to be used by our method. The resulting features are then represented by a 15-dimensional vector ($\mathbf{x}$). Each user is also represented by a set of features wich include gender, age and other features that compile geographical/regional information. As with the item features, we have created dummy variables for those categorical features, which resulted into a 85-dimensional feature vector ($\mathbf{t}$) for each user. As pointed out in the documentation of the dataset, Japanese food preferences are strongly correlated with geographical and regional information. Therefore, modeling user similarities may provide useful information during the elicitation process.

## 8.2    Evaluation Methodology and Experimental Details

We evaluate our method via 10-fold cross-validation, where we have sub-sampled the training folds in order to (a) keep the computational burden as low as possible and (b) show that we can learn sensible parameterizations based upon relatively low requirements in terms of the preferences seen on previous users. In particular, we have subsampled $50$ training users and selected about $5$ training pairwise preferences drawn from each of the $N = 10$ available items.

For the GPs we have used the squared exponential (SE) covariance functions with automatic relevance determination (ARD) for both $\kappa^t$ and $\kappa^x$ and have carried out hyperparameter learning via gradient-based optimization of the marginal likelihood in equation (28). We have initialized the hyper-parameters of the models deterministically, setting the signal variance and the length-scales of the covariance function to the initial values of $1$ and the $\sigma^2$ parameter to $0.01$.

In order to measure the quality of our preference elicitation approach we use the normalized loss as a function of the number of queries, where at each iteration the method provides a recommendation based on the available information. The normalized loss function is defined as: $(u^{best} - u^{pred})/u^{best}$, where $u^{best}$ is the best utility for a specific item/user and $u^{pred}$ is the utility achieved by the recommendation provided by the system.

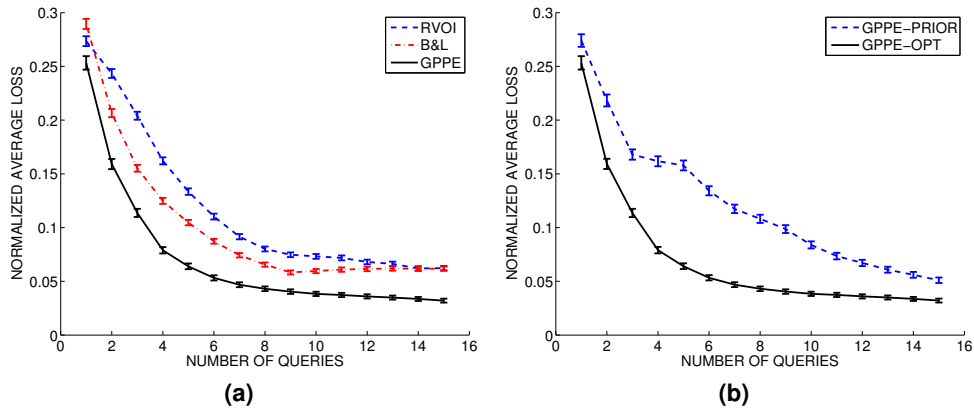

Figure 1: The Normalized average loss as a function of the number of queries with 2 standard (errors of the mean) error bars. (a) The performance of our model compared to the RVOI method described in [7] and the B&L heuristic over the full set of 5000 test users. (b) The performance of our model when the hyper-parameters have been optimized via maximization of the marginal likelihood (GPPE-OPT) compared to the same GP elicitation framework when these hyper-parameters have been set to their default values (GPPE-PRIOR).

We compare our approach to two baseline methods. One is the restricted value of information algorithm [7] and the other one is the best and largest heuristic, which we wil refer to as the RVOI method and the B&L heuristic respectively. The RVOI approach is also a VOI-based method but it does not leverage information from other users and it considers diagonal Gaussians as prior models of the latent utility functions. The B&L heuristic selects the current best item and the one with the largest uncertainty. Both baselines have been shown to be competitive methods for preference elicitation (see [7] for more details). Additionally, we compare our method when the hyper-parameters have been learned on the set of previously seen users with the same GP elicitation approach when the hyper-parameters have been set to the initial values described above. This allows us to show that, indeed, when prior information on user and item similarity is not available, our model does learn sensible settings of the hyper-parameters, which lead to better quality elicitation outcomes.

## 8.3 Results

Figure 1(a) shows the normalized average loss across all 5000 users as a function of the number of queries. As can be seen, on average, all competing methods reduce the expected loss as the number of queries increases. More importantly, our method (GPPE) clearly outperforms the other algorithms even for a small number of queries. This demonstrates that our approach exploits the inter-relations between users and items effectively in order to enhance the elicitation process on a new user. Although it may be surprising that the B&L heuristic outperforms the RVOI method, we point out that the evaluation of these methods presented in [7] did not consider real datasets as we do in our experiments.

Figure 1(b) shows the normalized average loss across all 5000 users for our method when the hyper-parameters have been set to the initial values described in section 8 (labeled in the figure as GPPE-PRIOR) and when the hyper-parameters have been optimized by maximization of the marginal likelihood on a set of previously seen users (labeled in the figure as GPPE-OPT). We can see that, indeed, the GPPE model that learns the hyper-parameters from previous users' data significantly outperforms the same method when these (hyper-)parameters are not optimized.

## 9   Related Work

Preference elicitation (PE) is an important component of recommender systems and market research. Traditional PE frameworks focus on modeling and eliciting a single user's preferences. We can categorize different PE frameworks in terms of query types. In [8], the authors propose to model

utilities as random variables, and refines utility uncertainty by using standard gamble queries. The same query type is also used in [9], which differs from [8] in treating PE as a Partially Observable Markov Decision Process (POMDP). However, standard gamble queries are difficult for users to respond to, and naturally lead to noisy responses. Simpler query types have also been used for PE. For example, [7] uses pairwise comparison queries, which are believed to have low cognitive load. Our work also adopts simple pairwise comparison queries, but it differs from [7] in that it makes use of users' preferences that have been seen before and does not assume additive independent utilities.

In the machine learning community preference learning has received substantial interest over the past few years. For example, one the most recent approaches to preference learning is presented in [10], where a multi-task learning approach to the problem of modeling human preferences is adopted by extending the model in [11] to deal with preference data. Their model follows a hierarchical approach based on *finite* Gaussian processes (GPs), where inter-user similarities are exploited by assuming that the subjects share a set of hyper-parameters. Their model is different to ours in that they consider the dual representation of the GPs as they do not generalize over user features. Furthermore, they do not address the elicitation problem, which is the main concern of this paper.

Extensions of the Gaussian process formalism to model ordinal data and user preferences are given in [12] and [13]. Both their prior and their likelihood models can be seen as single-user (task) specifications of our model. In other words, unlike the work of [10], their model (as ours) considers the function space view of the GPs but, unlike [10] and our approach, they do not address the multi-task case or generalize across users. More importantly, an elicitation framework for actively querying the user is not presented in such works.

[14] proposes an active preference learning method for discrete choice data. Their approach is based on the model in [13]. Unlike our approach they do not leverage information from seen preferences on previous users and hence their active preference learning process on a new user starts from scratch. This leads to the problem of either relying on good prior information on the covariance function or on hyper-parameter updating during the active learning process, which is computationally too expensive to be used in practice. Additionally, as their concern is on a possibly infinite set of discrete choices, their approach completely relies upon the expected improvement (EI) measure.

## 10    Conclusions & Future Work

In this paper we have presented a Gaussian process approach to the problem of preference elicitation. One of the crucial characteristics of our method is that it exploits user-similarity via a (correlated) Gaussian process prior over the users' latent utility functions. These similarities are "learned" from preferences on previous users. Our method maintains a flexible representation of the user's latent utility function, handles uncertainty in a principled manner and allows the incorporation of prior knowledge from different sources. The required *expected value of information* computations can be derived in closed-form to facilitate the selection of informative queries and determine the highest utility item from the available item set as quickly as possible.

We have shown the benefits of our method on a real dataset of 5000 users with preferences over 10 sushi types. In future work we aim at investigating other elicitation problems such as those involving a Likert scale [15] where our approach may be effective. The main practical constraint is that in order to carry out the evaluation (but not the application) of our method on real data we require the full set of preferences of the users over a set of items.

Our main motivation for the Laplace method is its computational efficiency. However, [10] has shown that this method is a good approximation to the posterior in the context of the preference learning problem. We intend to investigate other approximation methods to the posterior and marginal likelihood and their joint application with sparse approximation methods within our framework (see e.g. [16]), which will be required if the number of training users is large.

### Acknowledgments

NICTA is funded by the Australian Government as represented by the Department of Broadband, Communications and the Digital Economy and the Australian Research Council through the ICT Centre of Excellence program.

# References

[1] Donald R. Jones. A taxonomy of global optimization methods based on response surfaces. *Journal of Global Optimization*, 21(4):345–383, 2001.

[2] R.A. Howard. Information value theory. *IEEE Transactions on Systems Science and Cybernetics*, 2(1):22–26, 1966.

[3] Urszula Chajewska, Daphne Koller, and Ronald Parr. Making rational decisions using adaptive utility elicitation. In *Proceedings of the Seventeenth National Conference on Artificial Intelligence and Twelfth Conference on Innovative Applications of Artificial Intelligence*, pages 363–369. AAAI Press / The MIT Press, 2000.

[4] Vincent Conitzer. Eliciting single-peaked preferences using comparison queries. *Journal of Artificial Intelligence Research*, 35:161–191, 2009.

[5] Edwin V. Bonilla, Kian Ming A. Chai, and Christopher K. I. Williams. Multi-task Gaussian process prediction. In J.C. Platt, D. Koller, Y. Singer, and S. Roweis, editors, *Advances in Neural Information Processing Systems 20*, pages 153–160. MIT Press, Cambridge, MA, 2008.

[6] Toshihiro Kamishima. Nantonac collaborative filtering: recommendation based on order responses. In *Proceedings of the ninth ACM SIGKDD international conference on Knowledge discovery and data mining*, pages 583–588, New York, NY, USA, 2003. ACM.

[7] Shengbo Guo and Scott Sanner. Real-time multiattribute Bayesian preference elicitation with pairwise comparison queries. In *Proceedings of the Thirteenth International Conference on Artificial Intelligence and Statistics*, 2010.

[8] Urszula Chajewska and Daphne Koller. Utilities as random variables: Density estimation and structure discovery. In *Proceedings of the 16th Conference on Uncertainty in Artificial Intelligence*, pages 63–71. Morgan Kaufmann Publishers Inc., 2000.

[9] Craig Boutilier. A POMDP formulation of preference elicitation problems. In *Proceedings of the 18th National Conference on Artificial Intelligence*, pages 239–246, Menlo Park, CA, USA, 2002. American Association for Artificial Intelligence.

[10] Adriana Birlutiu, Perry Groot, and Tom Heskes. Multi-task preference learning with an application to hearing aid personalization. *Neurocomputing*, 73(7-9):1177–1185, 2010.

[11] Kai Yu, Volker Tresp, and Anton Schwaighofer. Learning Gaussian processes from multiple tasks. In *Proceedings of the 22nd international conference on Machine learning*, pages 1012–1019, New York, NY, USA, 2005. ACM.

[12] Wei Chu and Zoubin Ghahramani. Gaussian processes for ordinal regression. *Journal of Machine Learning Research*, 6:1019–1041, 2005.

[13] Wei Chu and Zoubin Ghahramani. Preference learning with Gaussian processes. In *Proceedings of the 22nd international conference on Machine learning*, pages 137–144, New York, NY, USA, 2005. ACM.

[14] Brochu Eric, Nando De Freitas, and Abhijeet Ghosh. Active preference learning with discrete choice data. In J.C. Platt, D. Koller, Y. Singer, and S. Roweis, editors, *Advances in Neural Information Processing Systems 20*, pages 409–416. MIT Press, Cambridge, MA, 2008.

[15] Rensis Likert. A technique for the measurement of attitudes. *Archives of Psychology*, 22(140):1–55, 1932.

[16] Joaquin Quiñonero Candela and Carl Edward Rasmussen. A unifying view of sparse approximate Gaussian process regression. *Journal of Machine Learning Research*, 6:1939–1959, 2005.

